# Hierarchical Penalization

**Marie Szafranski** [1]**, Yves Grandvalet** [1, 2] **and Pierre Morizet-Mahoudeaux** [1]
Heudiasyc [1], UMR CNRS 6599
Université de Technologie de Compiègne
BP 20529, 60205 Compiègne Cedex, France
IDIAP Research Institute [2]
Av. des Prés-Beudin 20
P.O. Box 592, 1920 Martigny, Switzerland
marie.szafranski@hds.utc.fr

## Abstract

*Hierarchical penalization* is a generic framework for incorporating prior information in the fitting of statistical models, when the explicative variables are organized in a hierarchical structure. The penalizer is a convex functional that performs soft selection at the group level, and shrinks variables within each group. This favors solutions with few leading terms in the final combination. The framework, originally derived for taking prior knowledge into account, is shown to be useful in linear regression, when several parameters are used to model the influence of one feature, or in kernel regression, for learning multiple kernels.

**Keywords −** *Optimization*: constrained and convex optimization. *Supervised learning:* regression, kernel methods, sparsity and feature selection.

## 1 Introduction

In regression, we want to explain or to predict a response variable $y$ from a set of explanatory variables $\boldsymbol{x} = (x^1, \dots, x^j, \dots, x^d)$, where $y \in \mathbb{R}$ and $\forall j, x^j \in \mathbb{R}$. For this purpose, we use a model such that $y = f(\boldsymbol{x}) + \epsilon$, where $f$ is a function able to characterize $y$ when $\boldsymbol{x}$ is observed and $\epsilon$ is a residual error.

Supervised learning consists in estimating $f$ from the available training dataset $S = \{(\boldsymbol{x}_i, y_i)\}_{i=1}^n$. It can be achieved in a predictive or a descriptive perspective: to predict accurate responses for future observations, or to show the correlations that exist between the set of explanatory variables and the response variable, and thus, give an interpretation to the model.

In the linear case, the function $f$ consists of an estimate $\boldsymbol{\beta} = (\beta_1, \dots, \beta_j, \dots, \beta_d)^{\text{t}}$ applied to $\boldsymbol{x}$, that is to say $f(\boldsymbol{x}) = \boldsymbol{x}\boldsymbol{\beta}$. In a predictive perspective, $\boldsymbol{x}\boldsymbol{\beta}$ produces an estimate of $y$, for any observation $\boldsymbol{x}$. In a descriptive perspective, $|\beta_j|$ can be interpreted as a degree of relevance of variable $x^j$.

*Ordinary Least Squares* (OLS) minimizes the sum of the residual squared error. When the explanatory variables are numerous and many of them are correlated, the variability of the OLS estimate tends to increase. This leads to reduced prediction accuracy, and an interpretation of the model becomes tricky.

*Coefficient shrinkage* is a major approach of regularization procedures in linear regression models. It overcomes the drawbacks described above by adding a constraint on the norm of the estimate $\boldsymbol{\beta}$. According to the chosen norm, coefficients associated to variables with little predictive information may be shrunk, or even removed when variables are irrelevant. This latest case is referred to as *variable selection*. In particular, *ridge regression* shrinks coefficients with regard to the $\ell_2$-norm, while the *lasso* (*Least Absolute Shrinkage and Selection Operator*) [1] and the *lars* (*Least Angle Regression Stepwise*) [2] both shrink and remove coefficients using the $\ell_1$-norm.

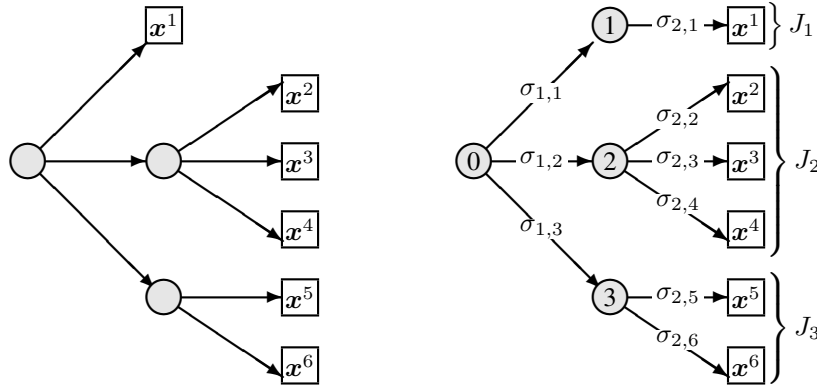

Figure 1: left: toy-example of the original structure of variables; right: equivalent tree structure considered for the formalization of the scaling problem.

In some applications, explanatory variables that share a similar characteristic can be gathered into groups – or *factors*. Sometimes, they can be organized hierarchically. For instance, in genomics, where explanatory variables are (products of) genes, some factors can be identified from the prior information available in the hierarchies of *Gene Ontology*. Then, it becomes necessary to find methods that retain meaningful factors instead of individual variables.

*Group-lasso* and *group-lars* [3] can be considered as hierarchical penalization methods, with trees of height two defining the hierarchies. They perform variable selection by encouraging sparseness over predefined factors. These techniques seem perfectible in the sense that hierarchies can be extended to more than two levels and sparseness integrated within groups. This papers proposes a penalizer, derived from an adaptive penalization formulation [4], that highlights factors of interest by balancing constraints on each element, at each level of a hierarchy. It performs soft selection at the factor level, and shrinks variables within groups, to favor solutions with few leading terms.

Section 2 introduces the framework of *hierarchical penalization* and the associated algorithm is presented in Section 3. Section 4 shows how this framework can be applied to linear and kernel regression. We conclude with a general survey of our future works.

## 2 Hierarchical Penalization

### 2.1 Formalization

We introduce *hierarchical penalization* by considering problems where the variables are organized in a tree structure of height two, such as the example displayed in figure 1. The nodes of height one are labelled in $\{1, \ldots, K\}$. The set of children (that is, leaves) of node $k$ is denoted $J_k$ and its cardinality is $d_k$. As displayed on the right-hand-side of figure 1, a branch stemming from the root and going to node $k$ is labelled by $\sigma_{1,k}$, and the branch reaching leaf $j$ is labelled by $\sigma_{2,j}$.

We consider the problem of minimizing a differentiable loss function $L(\cdot)$, subject to sparseness constraints on $\boldsymbol{\beta}$ and the subsets of $\boldsymbol{\beta}$ defined in a tree hierarchy. This reads

$$
\begin{cases}
\min_{\boldsymbol{\beta}, \boldsymbol{\sigma}} & L(\boldsymbol{\beta}) + \lambda \sum_{k=1}^{K} \sum_{j \in J_k} \frac{\beta_j^2}{\sqrt{\sigma_{1,k}\, \sigma_{2,j}}} \ , & \text{(1a)} \\[2ex]
\text{subject to} & \sum_{k=1}^{K} d_k\, \sigma_{1,k} = 1 \ , \qquad\qquad \sum_{j=1}^{d} \sigma_{2,j} = 1 \ , & \text{(1b)} \\[2ex]
& \sigma_{1,k} \geq 0 \quad k = 1, \ldots, K \ , \qquad \sigma_{2,j} \geq 0 \quad j = 1, \ldots, d \ , & \text{(1c)}
\end{cases}
$$

where $\lambda > 0$ is a Lagrangian parameter that controls the amount of shrinkage, $x/y$ is defined by continuation at zero as $x/0 = \infty$ if $x \neq 0$ and $0/0 = 0$.

The second term of expression (1a) penalizes $\boldsymbol{\beta}$, according to the tree structure, via scaling factors $\boldsymbol{\sigma}_1$ and $\boldsymbol{\sigma}_2$. The constraints (1b) shrink the coefficients $\boldsymbol{\beta}$ at group level and inside groups. In what follows, we show that problem (1) is convex and that this joint shrinkage encourages sparsity at the group level.

## 2.2  Two important properties

We first prove that the optimization problem (1) is tractable and moreover convex. Then, we show an equivalence with another optimization problem, which exhibits the exact nature of the constraints applied to the coefficients $\boldsymbol{\beta}$.

**Proposition 1** *Provided $L(\cdot)$ is convex, problem (1) is convex.*

**Proof:** *A problem minimizing a convex criterion on a convex set is convex. Since $L(\cdot)$ is convex and $\lambda$ is positive, the criterion (1a) is convex provided $f(x,y,z) = \frac{x^2}{\sqrt{yz}}$ is convex. To show this, we compute the Hessian:*

$$4(yz)^{\frac{1}{2}}\nabla^2 f(x,y,z) = \begin{bmatrix} 8 & -4\frac{x}{y} & -4\frac{x}{z} \\ -4\frac{x}{y} & 3\frac{x^2}{y^2} & \frac{x^2}{yz} \\ -4\frac{x}{z} & \frac{x^2}{yz} & 3\frac{x^2}{z^2} \end{bmatrix} = 2\begin{bmatrix} 2 \\ -\frac{x}{y} \\ -\frac{x}{z} \end{bmatrix}\begin{bmatrix} 2 \\ -\frac{x}{y} \\ -\frac{x}{z} \end{bmatrix}^{\mathrm{t}} + \begin{bmatrix} 0 \\ \frac{x}{y} \\ -\frac{x}{z} \end{bmatrix}\begin{bmatrix} 0 \\ \frac{x}{y} \\ -\frac{x}{z} \end{bmatrix}^{\mathrm{t}} \quad .$$

*Hence, the Hessian is positive semi-definite, and criterion (1a) is convex.*

*Next, constraints (1c) define half-spaces for $\boldsymbol{\sigma}_1$ and $\boldsymbol{\sigma}_2$, which are convex sets. Equality constraints (1b) define linear subspaces of dimension $K-1$ and $d-1$ which are also convex sets. The intersection of convex sets being a convex set, the constraints define a convex admissible set, and problem (1) is convex.* ☐

**Proposition 2** *Problem (1) is equivalent to*

$$\min_{\boldsymbol{\beta}} L(\boldsymbol{\beta}) + \lambda \left( \sum_{k=1}^{K} d_k^{\frac{1}{4}} \left( \sum_{j\in J_k} |\beta_j|^{\frac{4}{3}} \right)^{\frac{3}{4}} \right)^2 \quad . \tag{2}$$

**Sketch of proof:**

*The Lagrangian of problem (1) is*

$$\begin{aligned} \mathcal{L} &= L(\boldsymbol{\beta}) + \lambda \sum_{k=1}^{K}\sum_{j\in J_k} \frac{\beta_j^2}{\sqrt{\sigma_{1,k}\,\sigma_{2,j}}} + \nu_1\left(\sum_{k=1}^{K} d_k\,\sigma_{1,k} - 1\right) + \\ &\quad \nu_2\left(\sum_{j=1}^{d}\sigma_{2,j} - 1\right) - \sum_{k=1}^{K}\xi_{1,k}\,\sigma_{1,k} - \sum_{j=1}^{d}\xi_{2,j}\,\sigma_{2,j} \quad . \end{aligned}$$

*Hence, the optimality conditions for $\sigma_{1,k}$ and $\sigma_{2,j}$ are*

$$\begin{cases} \dfrac{\partial \mathcal{L}}{\partial \sigma_{1,k}} = 0 \\[2mm] \dfrac{\partial \mathcal{L}}{\partial \sigma_{2,j}} = 0 \end{cases} \Rightarrow \begin{cases} -\dfrac{\lambda}{2}\sum_{j\in J_k}\dfrac{\beta_j^2}{\sigma_{1,k}^{\frac{3}{2}}\sigma_{2,j}^{\frac{1}{2}}} + \nu_1 d_k - \xi_{1,k} = 0 \\[3mm] -\dfrac{\lambda}{2}\dfrac{\beta_j^2}{\sigma_{1,k}^{\frac{1}{2}}\sigma_{2,j}^{\frac{3}{2}}} + \nu_2 - \xi_{2,j} = 0 \end{cases} \quad .$$

*After some tedious algebra, the optimality conditions for $\sigma_{1,k}$ and $\sigma_{2,j}$ can be expressed as*

$$\sigma_{1,k} = \frac{d_k^{-\frac{3}{4}}\,(s_k)^{\frac{3}{4}}}{\sum_{k=1}^{K} d_k^{\frac{1}{4}}\,(s_k)^{\frac{3}{4}}} \quad , \quad \text{and} \quad \sigma_{2,j} = \frac{d_k^{\frac{1}{4}}\,|\beta_j|^{\frac{4}{3}}}{(s_k)^{\frac{1}{4}}\sum_{k=1}^{K} d_k^{\frac{1}{4}}\,(s_k)^{\frac{3}{4}}} \quad \text{for } j\in J_k \quad ,$$

*where $s_k = \sum_{j\in J_k} |\beta_j|^{\frac{4}{3}}$. Plugging these conditions in criterion (1a) yields the claimed result.* ☐

## 2.3 Sparseness

Proposition 2 shows how the penalization influences the groups of variables and each variable in each group. Note that, thanks to the positivity of the squared term in (2), the expression can be further simplified to

$$\min_{\boldsymbol{\beta}} L(\boldsymbol{\beta}) + \nu \sum_{k=1}^{K} d_k^{\frac{1}{4}} \left( \sum_{j \in J_k} |\beta_j|^{\frac{4}{3}} \right)^{\frac{3}{4}} , \qquad (3)$$

where, for any $L(\boldsymbol{\beta})$, there is a one-to-one mapping from $\lambda$ in (2) to $\nu$ in (3). This expression can be interpreted as the Lagrangian formulation of a constrained optimization problem, where the admissible set for $\boldsymbol{\beta}$ is defined by the multiplicand of $\nu$.

We display the shape of the admissible set in figure 2, and compare it to *ridge regression*, which does not favor sparsity, *lasso*, which encourages sparsity for all variables but does not take into account the group structure, and *group-lasso*, which is invariant to rotations of within-group variables. One sees that *hierarchical penalization* combines some features of *lasso* and *group-lasso*.

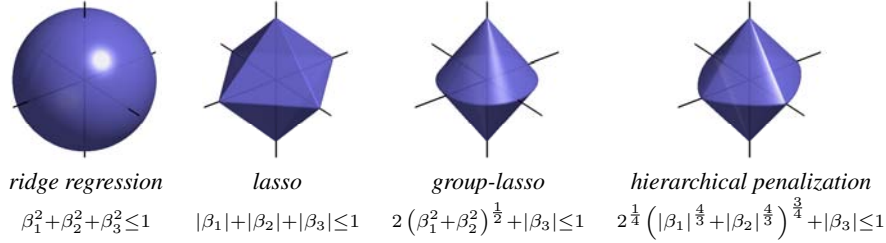

| *ridge regression* | *lasso* | *group-lasso* | *hierarchical penalization* |
|---|---|---|---|
| $\beta_1^2 + \beta_2^2 + \beta_3^2 \le 1$ | $|\beta_1| + |\beta_2| + |\beta_3| \le 1$ | $2\left(\beta_1^2 + \beta_2^2\right)^{\frac{1}{2}} + |\beta_3| \le 1$ | $2^{\frac{1}{4}}\left(|\beta_1|^{\frac{4}{3}} + |\beta_2|^{\frac{4}{3}}\right)^{\frac{3}{4}} + |\beta_3| \le 1$ |

Figure 2: Admissible sets for various penalties, the two horizontal axes are the $(\beta_1, \beta_2)$ plane (first group) and the vertical axis is for $\beta_3$ (second group).

By looking at the curvature of these sets when they meet axes, one gets a good intuition on why *ridge regression* does not suppress variables, why *lasso* does, why *group-lasso* suppresses groups of variables but not within-group variables, and why *hierarchical penalization* should do both. This intuition is however not correct for *hierarchical penalization* because the boundary of the admissible set is differentiable in the within-group hyper-plane $(\beta_1, \beta_2)$ at $\beta_1 = 0$ and $\beta_2 = 0$. However, as its curvature is very high, solutions with few leading terms in the within-group variables are encouraged.

To go beyond the hints provided by these figures, we detail here the optimality conditions for $\boldsymbol{\beta}$ minimizing (3). The first-order optimality conditions are

1. for $\beta_j = 0$, $j \in J_k$ and $\sum_{j \in J_k} |\beta_j| = 0$, $\dfrac{\partial L(\boldsymbol{\beta})}{\partial \beta_j} + \nu\, d_k^{\frac{1}{4}} v_j = 0$, where $v_j \in [-1, 1]$;

2. for $\beta_j = 0$, $j \in J_k$ and $\sum_{j \in J_k} |\beta_j| \ne 0$, $\dfrac{\partial L(\boldsymbol{\beta})}{\partial \beta_j} = 0$;

3. for $\beta_j \ne 0$, $j \in J_k$, $\dfrac{\partial L(\boldsymbol{\beta})}{\partial \beta_j} + \nu\, d_k^{\frac{1}{4}} \operatorname{sign}(\beta_j)\left(1 + \dfrac{1}{|\beta_j|^{\frac{4}{3}}} \sum_{\substack{\ell \in J_k \\ \ell \ne j}} |\beta_\ell|^{\frac{4}{3}}\right)^{-\frac{1}{4}} = 0$.

These equations signify respectively that

1. the variables belonging to groups that are estimated to be irrelevant are penalized with the highest strength, thus limiting the number of groups influencing the solution;

2. when a group has some non-zero relevance, all variables enter the set of active variables provided they influence the fitting criterion;

3. however, the penalization strength increases very rapidly (as a smooth step function) for small values of $|\beta_j|$, thus limiting the number of $\beta_j$ with large magnitude.

Overall, *hierarchical penalization* is thus expected to provide solutions with few active groups and few leading variables within each group.

## 3 Algorithm

To solve problem (3), we use an active set algorithm, based on the approach proposed by Osborne et al. [5] for the *lasso*. This algorithm iterates two phases: first, the optimization problem is solved with a sub-optimal set of active variables, that is, non-zero variables: we define $\mathcal{A} = \{j \,|\, \beta_j \neq 0\}$, the current active set of variables, $\boldsymbol{\gamma} = \{\beta_j\}_{j \in \mathcal{A}}$, the vector of coefficients associated to $\mathcal{A}$, and $G_k = \{J_k \cap \mathcal{A}\}$, the subset of coefficients $\boldsymbol{\gamma}$ associated to group $k$. Then, at each iteration, we solve the problem

$$\min_{\boldsymbol{\gamma}} \mathcal{L}(\boldsymbol{\gamma}) = L(\boldsymbol{\gamma}) + \nu \sum_{k=1}^{K} d_k^{\frac{1}{4}} \left( \sum_{j \in G_k} |\gamma_j|^{\frac{4}{3}} \right)^{\frac{3}{4}}, \tag{4}$$

by alternating steps A and B described below. Second, the set of active variables is incrementally updated as detailed in steps C and D.

A  *Compute a candidate update from an admissible vector $\boldsymbol{\gamma}$*

   The goal is to solve $\min_{\mathbf{h}} \mathcal{L}(\boldsymbol{\gamma} + \mathbf{h})$, where $\boldsymbol{\gamma}$ is the current estimate of the solution and $\mathbf{h} \in \mathbb{R}^{|\mathcal{A}|}$. The difficulties in solving (4) stem from the discontinuities of the derivative due to the absolute values. These difficulties are circumvented by replacing $|\gamma_j + h_j|$ by $\text{sign}(\gamma_j)(\gamma_j + h_j)$. This enables the use of powerful continuous optimizers based either on the Newton, quasi-Newton or conjugate gradient methods according to the size of the problem.

B  *Obtain a new admissible vector $\boldsymbol{\gamma}^\dagger$*

   Let $\boldsymbol{\gamma}^\dagger = \boldsymbol{\gamma} + \mathbf{h}$. If for all $j$, $\text{sign}(\gamma_j^\dagger) = \text{sign}(\gamma_j)$, then $\boldsymbol{\gamma}$ is sign-feasible, and we go to step C, otherwise:

   B.1  Let $\mathcal{S}$ be the set of indices $m$ such that $\text{sign}(\gamma_m^+) \neq \text{sign}(\gamma_m)$. Let $\mu = \min_{m \in \mathcal{S}} -\frac{\gamma_m}{h_m}$, that is, $\mu$ is the largest step in direction $\mathbf{h}$ such that $\text{sign}(\gamma_m + \mu h_m) = \text{sign}(\gamma_m)$, except for one variable, $\ell = \arg \min_m -\frac{\gamma_m}{h_m}$, for which $\gamma_\ell + \mu h_\ell = 0$.

   B.2  Set $\boldsymbol{\gamma} = \boldsymbol{\gamma} + \mu \mathbf{h}$ and $\text{sign}(\gamma_\ell) = -\text{sign}(\gamma_\ell)$, and compute a new direction $\mathbf{h}$ as in step A. If, for the new solution $\boldsymbol{\gamma}^\dagger$, $\text{sign}(\gamma_\ell^\dagger) \neq \text{sign}(\gamma_\ell)$, then $\ell$ is removed from $\mathcal{A}$. Go to step A.

   B.3  Iterate step B until $\boldsymbol{\gamma}$ is sign-feasible.

C  *Test optimality of $\boldsymbol{\gamma}$*

   If the appropriate optimality condition holds for all inactive variables $\beta_\ell$ $(\beta_\ell = 0)$, that is

   C.1  for $\ell \in J_k$, where $\sum_{j \in J_k} |\beta_j| = 0$, then $\left| \frac{\partial L(\boldsymbol{\beta})}{\partial \beta_\ell} \right| \leq \nu \, d_k^{\frac{1}{4}}$,

   C.2  for $\ell \in J_k$, where $\sum_{j \in J_k} |\beta_j| \neq 0$, then $\frac{\partial L(\boldsymbol{\beta})}{\partial \beta_\ell} = 0$,

   then $\boldsymbol{\gamma}$ is a solution. Else, go to step D.

D  *Select the variable that enters the active set*

   D.1  Select variable $\ell$, $\ell \notin \mathcal{A}$ that maximizes $d_k^{-\frac{1}{4}} \left| \frac{\partial L(\boldsymbol{\beta})}{\partial \beta_\ell} \right|$, where $k$ is the group of variable $\ell$.

   D.2  Update the active set: $\mathcal{A} \leftarrow \mathcal{A} \cup \{\ell\}$, with initial vector: $\boldsymbol{\gamma} = [\boldsymbol{\gamma}, 0]^{\text{t}}$ where the sign of the new zero component is $-\text{sign}\left( \frac{\partial L(\boldsymbol{\beta})}{\partial \beta_\ell} \right)$.

   D.3  Go to step A.

The algorithm is initialized with $\mathcal{A} = \emptyset$, and the first variable is selected with the process described at step D.

# 4 Experiments

We illustrate on two datasets how *hierarchical penalization* can be useful in exploratory analysis and in prediction. Then, we show how the algorithm can be applied for multiple kernel learning in kernel regression.

## 4.1 Abalone Database

The Abalone problem [6] consists in predicting the age of abalone from physical measurements. The dataset is composed of 8 attributes. One concerns the sex of abalone, and has been encoded with dummy variables, that is $x_i^{\text{sex}} = (100)$ for male, $x_i^{\text{sex}} = (010)$ for female, or $x_i^{\text{sex}} = (001)$ for infant. This variable defines the first group. The second group is composed of 3 attributes concerning size parameters (length, diameter and height), and the last group is composed of weight parameters (whole, shucked, viscera and shell weight).

We randomly selected 2920 examples for training, including the tuning of $\nu$ by 10-fold cross validation, and left the 1257 other for testing. The mean squared test error is at par with *lasso* (4.3). The coefficients estimated on the training set are reported in table 4.1. Weight parameters are a main contributor to the estimation of the age of an abalon, while sex is not essential, except for infant.

| | | | | | |
|---|---|---|---|---|---|
| sex | 0.051 | 0.036 | -0.360 | | 0.516 |
| size | -0.044 | 1.134 | 0.358 | | 1.7405 |
| weight | 4.370 | -4.499 | -1.110 | 1.399 | 11.989 |

Table 1: Coefficients obtained on the Abalone dataset. The last column represents the value $d_k^{\frac{1}{4}} \left( \sum_{j \in J_k} |\beta_j|^{\frac{4}{3}} \right)^{\frac{3}{4}}$.

## 4.2 Delve Census Database

The Delve Census problem [7] consists in predicting the median price of a house in different survey regions. Each 22732 survey region is represented by 134 demographic information measurements. Several prototypes are available. We focussed on the prototype "house-price-16L", composed of 16 variables. We derived this prototype by including all the other variables related to these 16 variables. The final dataset is then composed of 37 variables, split up into 10 groups[1].

We randomly selected 8000 observations for training and left the 14732 for testing. We divided the training observations into 10 distinct datasets. For each dataset, the parameter $\nu$ was selected by a 10-fold cross validation, and the mean squared error was computed on the testing set. We reported on table 4.2 the mean squared test errors obtained with the *hierarchical penalization* (hp), the *group-lasso* (gl) and the *lasso* estimates.

| Datasets | 1 | 2 | 3 | 4 | 5 | 6 | 7 | 8 | 9 | 10 | mean error |
|---|---|---|---|---|---|---|---|---|---|---|---|
| hp ($\times 10^9$) | **2.363** | 2.745 | **2.289** | **4.481** | **2.211** | **2.364** | **2.460** | 2.298 | 2.461 | **2.286** | 2.596 |
| gl ($\times 10^9$) | 2.429 | **2.460** | **2.289** | 4.653 | 2.230 | **2.364** | 2.472 | 2.308 | **2.454** | 2.291 | **2.595** |
| lasso ($\times 10^9$) | 2.380 | 2.716 | 2.293 | 4.656 | 2.216 | 2.368 | 2.490 | **2.295** | 2.483 | 2.288 | 2.618 |

Table 2: Mean squared test errors obtained with different methods for the 10 datasets.

*Hierarchical penalization* performs better than *lasso* on 8 datasets. It also performs better than *group-lasso* on 6 datasets, and obtains equal results on 2 datasets. However the lowest overall mean error is achieved by *group-lasso*.

## 4.3 Multiple Kernel Learning

Multiple Kernel Learning has drawn much interest in classification with support vector machines (SVMs) starting from the work of Lanckriet et al. [8]. The problem consists in learning a convex

combination of kernels in the SVM optimization algorithm. Here, we show that *hierarchical penalization* is well suited for this purpose for other kernel predictors, and we illustrate its effect on kernel smoothing in the regression setup.

Kernel smoothing has been studied in nonparametric statistics since the 60's [9]. Here, we consider the model where the response variable $y$ is estimated by a sum of kernel functions

$$y_i = \sum_{j=1}^{n} \beta_j \, \kappa_h(\boldsymbol{x}_i, \boldsymbol{x}_j) + \epsilon_i \;\;,$$

where $\kappa_h$ is the kernel with scale factor (or bandwidth) $h$, and $\epsilon_i$ is a residual error. For the purpose of combining $K$ bandwidths, the general criterion (3) reads

$$\min_{\{\boldsymbol{\beta}_k\}_{k=1}^{K}} \sum_{i=1}^{n} \left( y_i - \sum_{k=1}^{K} \sum_{j=1}^{n} \beta_{k,j} \, \kappa_{h_k}(\boldsymbol{x}_i, \boldsymbol{x}_j) \right)^2 + \nu \sum_{k=1}^{K} n_k^{\frac{1}{4}} \left( \sum_{j=1}^{n} |\beta_{k,j}|^{\frac{4}{3}} \right)^{\frac{3}{4}} \;\;. \tag{5}$$

The penalized model (5) has been applied to the motorcycle dataset [9]. This uni-dimensional problems enables to display the contribution of each bandwidth to the solution. We used Gaussian kernels, with 7 bandwidths ranging from $10^{-1}$ to $10^2$.

Figure 3 displays the results obtained for different penalization parameters: the estimated function obtained by the combination of the selected bandwidths, and the contribution of each bandwidth to the model. We display three settings for the penalization parameter $\nu$, corresponding to slight over-fitting, good fit and slight under-fitting. The coefficients of bandwidths $h_2$, $h_6$ and $h_7$ were always null and are thus not displayed. As expected, when the penalization parameter $\nu$ increases, the fit becomes smoother, and the number of contributing bandwidths decreases. We also observe that the effective contribution of some bandwidths is limited to a few kernels: there are few leading terms in the expansion.

## 5   Conclusion and further works

*Hierarchical penalization* is a generic framework enabling to process hierarchically structured variables by usual statistical models. The structure is provided to the model via constraints on the subgroups of variables defined at each level of the hierarchy. The fitted model is then biased towards statistical explanations that are "simple" with respect to this structure, that is, solutions which promote a small number of groups of variables, with a few leading components.

In this paper, we detailed the general framework of *hierarchical penalization* for tree structures of height two, and discussed its specific properties in terms of convexity and parsimony. Then, we proposed an efficient active set algorithm that incrementally builds an optimal solution to the problem. We finally illustrated how the approach can be used when groups of features, or when discrete variables exist, after being encoded by several binary variables, result in groups of variables. Finally, we also shown how the algorithm can be used to learn from multiple kernels in regression. We are now performing quantitative empirical evaluations, with applications to regression, classification and clustering, and comparisons to other regularization schemes, such as the *group-lasso*.

We then plan to extend the formalization to hierarchies of arbitrary height, whose properties are currently under study. We will then be able to tackle new applications, such as genomics, where the available gene ontologies are hierarchical structures that can be faithfully approximated by trees.

## Footnotes

[1] A description of the dataset is available at http://www.hds.utc.fr/~mszafran/nips07/.

## References

[1] R. Tibshirani. Regression shrinkage and selection via the lasso. *Journal of the Royal Statistical Society. Series B*, 58(1):267–288, 1996.

[2] B. Efron, T. Hastie, I. Johnstone, and R. Tibshirani. Least angle regression. *Annals of Statistics*, 32(2):407–499, 2004.

[3] M. Yuan and Y. Lin. Model selection and estimation in regression with grouped variables. *Journal of the Royal Statistical Society. Series B*, 68(1):49–67, 2006.

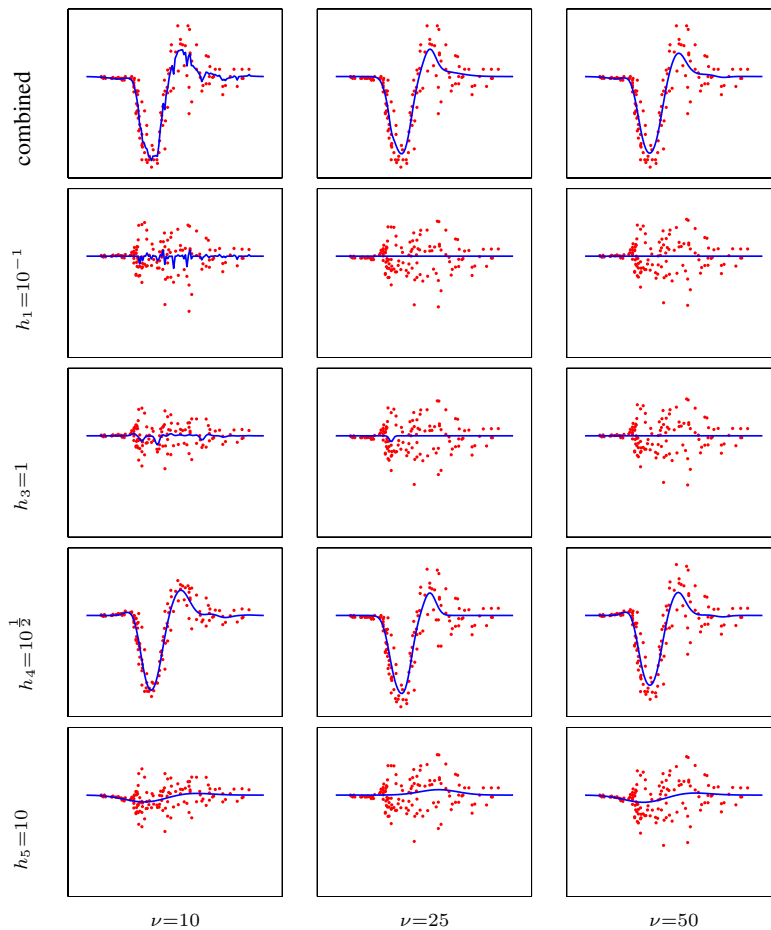

Figure 3: *Hierarchical penalization* applied to kernel smoothing on the motorcycle data. Combined: the points represent data and the solid line the function of estimated responses. Isolated bandwidths: the points represent partial residuals and the solid line represents the contribution of the bandwidth to the model.

[4] Y. Grandvalet and S. Canu. Adaptive scaling for feature selection in SVMs. In *Advances in Neural Information Processing Systems*, volume 15. MIT Press, 2003.

[5] M. R. Osborne, B. Presnell, and B. A. Turlach. On the lasso and its dual. *Journal of Computational and Graphical Statistics*, 9(2):319–337, June 2000.

[6] C.L. Blake D.J. Newman, S. Hettich and C.J. Merz. UCI repository of machine learning databases, 1998. URL http://www.ics.uci.edu/~mlearn/MLRepository.html.

[7] Delve: Data for evaluating learning in valid experiments. URL http://www.cs.toronto.edu/~delve/.

[8] G. Lanckriet, T. De Bie, N. Cristianini, M. Jordan, and W. Noble. A statistical framework for genomic data fusion. *Bioinformatics*, 20:2626–2635, 2004.

[9] W. Härdle. *Applied Nonparametric Regression*, volume 19. Economic Society Monographs, 1990.

